# Convergence and Rate of Convergence of A Manifold-Based Dimension Reduction Algorithm

**Andrew K. Smith,    Xiaoming Huo**
School of Industrial and Systems Engineering
Georgia Institute of Technology
Atlanta, GA 30332
andrewsmith81@gmail.com, huo@gatech.edu

**Hongyuan Zha**
College of Computing
Georgia Institute of Technology
Atlanta, GA 30332
zha@cc.gatech.edu

## Abstract

We study the convergence and the rate of convergence of a local manifold learning algorithm: LTSA [13]. The main technical tool is the perturbation analysis on the linear invariant subspace that corresponds to the solution of LTSA. We derive a worst-case upper bound of errors for LTSA which naturally leads to a convergence result. We then derive the rate of convergence for LTSA in a special case.

## 1   Introduction

Manifold learning (ML) methods have attracted substantial attention due to their demonstrated potential. Many algorithms have been proposed and some work has appeared to analyze the performance of these methods. The main contribution of this paper is to establish some asymptotic properties of a local manifold learning algorithm: LTSA [13], as well as a demonstration of some of its limitations. The key idea in the analysis is to treat the solutions computed by LTSA as invariant subspaces of certain matrices, and then carry out a matrix perturbation analysis.

Many efficient ML algorithms have been developed including locally linear embedding (LLE) [6], ISOMAP [9], charting [2], local tangent space alignment (LTSA) [13], Laplacian eigenmaps [1], and Hessian eigenmaps [3]. A common feature of many of these manifold learning algorithms is that their solutions correspond to invariant subspaces, typically the eigenspace associated with the smallest eigenvalues of a kernel or alignment matrix. The exact form of this matrix, of course, depends on the details of the particular algorithm.

We start with LTSA for several reasons. First of all, in numerical simulations (e.g., using the tools offered by [10]), we find empirically that LTSA performs among the best of the available algorithms. Second, the solution to each step of the LTSA algorithm is an invariant subspace, which makes analysis of its performance more tractable. Third, the similarity between LTSA and several other ML algorithms (e.g., LLE, Laplacian eigenmaps and Hessian eigenmaps) suggests that our results may generalize. Our hope is that this performance analysis will provide a theoretical foundation for the application of ML algorithms.

The rest of the paper is organized as follows. The problem formulation and background information are presented in Section 2. Perturbation analysis is carried out, and the main theorem is proved (Theorem 3.7) in Section 3. Rate of convergence under a special case is derived in Section 4. Some discussions related to existing work in this area are included in Section 5. Finally, we present concluding remarks in Section 6.

## 2 Manifold Learning and LTSA

We formulate the manifold learning problem as follows. For a positive integer $n$, let $y_i \in I\!\!R^D, i = 1, 2, \ldots, n$, denote $n$ observations. We assume that there is a mapping $f : I\!\!R^d \to I\!\!R^D$ which satisfies a set of regularity conditions (detailed in the next subsection). In addition, we require another set of (possibly multivariate) values $x_i \in I\!\!R^d, d < D, i = 1, 2, \ldots, n$, such that

$$y_i = f(x_i) + \varepsilon_i, \quad i = 1, 2, \ldots, n, \tag{1}$$

where $\varepsilon_i \in I\!\!R^D$ denotes a random error. For example, we may assume $\varepsilon_i \sim N(0, \sigma^2 I_D)$; i.e., a multivariate normal distribution with mean zero and variance-covariance proportional to the identity matrix. The central questions of manifold learning are: 1) Can we find a set of low-dimensional vectors such that equation (1) holds? 2) What kind of regularity conditions should be imposed on $f$? 3) Is the model well defined? These questions are the main focus of this paper.

### 2.1 A Pedagogical Example

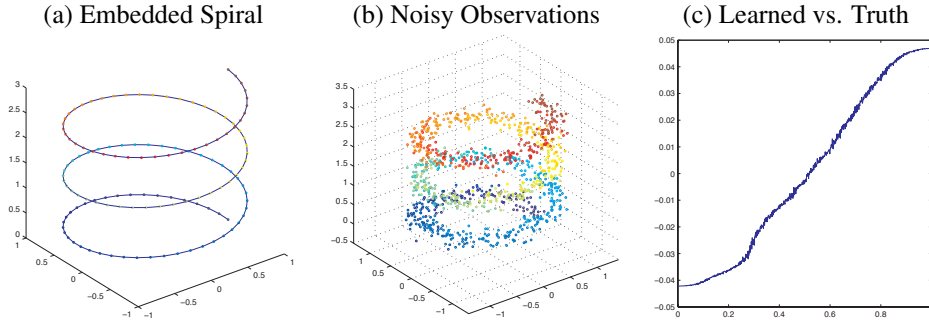

Figure 1: An illustrative example of LTSA in nonparametric dimension reduction. The straight line pattern in (c) indicates that the underlying parametrization has been approximately recovered.

An illustrative example of dimension reduction that makes our formulation more concrete is given in Figure 1. Subfigure (a) shows the true underlying structure of a toy example, a 1-D spiral. The *noiseless* observations are equally spaced points on this spiral. In subfigure (b), $1024$ *noisy* observations are generated with multivariate noise satisfying $\varepsilon_i \sim N(0, \frac{1}{100}\mathbf{I}_3)$. We then apply LTSA to the noisy observations, using $k = 10$ nearest neighbors. In subfigure (c), the result from LTSA is compared with the true parametrization. When the underlying parameter is faithfully recovered, one should see a straight line, which is observed to hold approximately in subfigure (c).

### 2.2 Regularity and Uniqueness of the Mapping $f$

If the conditions on the mapping $f$ are too general, the model in equation (1) is not well defined. For example, if the mapping $f(\cdot)$ and point set $\{x_i\}$ satisfy (1), so do $f(A^{-1}(\cdot - b))$ and $\{Ax_i + b\}$, where $A$ is an invertible $d$ by $d$ matrix and $b$ is a $d$-dimensional vector. As being common in the manifold-learning literature, we adopt the following condition on $f$.

**Condition 2.1 (Local Isometry)** *The mapping $f$ is locally isometric: For any $\varepsilon > 0$ and $x$ in the domain of $f$, let $N_\varepsilon(x) = \{z : \|z - x\|_2 < \varepsilon\}$ denote an $\varepsilon$-neighborhood of $x$ using Euclidean distance. We have*

$$\|f(x) - f(x_0)\|_2 = \|x - x_0\|_2 + o(\|x - x_0\|_2).$$

The above condition indicates that in a local sense, $f$ preserves Euclidean distance. Let $J(f; x_0)$ denote the Jacobian of $f$ at $x_0$. We have $J(f; x_0) \in I\!\!R^{D \times d}$, where each column (resp., row) of $J(f; x_0)$ corresponds to a coordinate in the feature (resp., data) space. The above in fact implies the following lemma [13].

**Lemma 2.2** *The matrix $J(f; x_0)$ is orthonormal for any $x_0$, i.e., $J^T(f; x_0)J(f; x_0) = I_d$.*

Given the previous condition, model (1) is still not uniquely defined. For example, for any $d$ by $d$ orthogonal matrix $O$ and any $d$-dimensional vector $b$, if $f(\cdot)$ and $\{x_i\}$ satisfy (1) and Condition 2.1, so do $f(O^T(\cdot - b))$ and $\{Ox_i + b\}$. We can force $b$ to be $0$ by imposing the condition that $\sum_i x_i = 0$. In dimension reduction, we can consider the sets $\{x_i\}$ and $\{Ox_i\}$ "invariant," because one is just a rotation of the other. In fact, the invariance coincides with the concept of "invariant subspace" to be discussed.

**Condition 2.3 (Local Linear Independence Condition)** *Let $Y_i \in I\!\!R^{D \times k}$, $1 \leq i \leq n$, denote a matrix whose columns are made by the $i$th observation $y_i$ and its $k - 1$ nearest neighbors. We choose $k - 1$ neighbors so that the matrix $Y_i$ has $k$ columns. It is generally assumed that $d < k$. For any $1 \leq i \leq n$, the rank of $Y_i \overline{P}_k$ is at least $d$; in other words, the $d$th largest singular value of matrix $Y_i \overline{P}_k$ is greater than $0$.*

In the above, we use the projection matrix $\overline{P}_k = I_k - \frac{1}{k} \cdot \mathbf{1}_k \mathbf{1}_k^T$, where $I_k$ is the $k$ by $k$ identity matrix and $\mathbf{1}_k$ is a $k$-dimensional column vector of ones. The regularity of the manifold can be determined by the Hessians of the mapping. Rewrite $f(x)$ for $x \in I\!\!R^d$ as $f(x) = (f_1(x), f_2(x), \ldots, f_D(x))^T$. Furthermore, let $x = (x_1, \ldots, x_d)^T$. The Hessian is a $D$ by $D$ matrix,

$$[H_i(f; x)]_{jk} = \frac{\partial^2 f_i(x)}{\partial x_j \partial x_k}, \quad 1 \leq i \leq D, 1 \leq j, k \leq d.$$

The following condition ensures that $f$ is locally smooth. We impose a bound on all the components of the Hessians.

**Condition 2.4 (Regularity of the Manifold)** *$|[H_i(f; x)]_{jk}| \leq C_1$ for all $i, j$, and $k$, where $C_1 > 0$ is a prescribed constant.*

## 2.3 Solutions as Invariant Subspaces and a Related Metric

We now give a more detailed discussion of invariant subspaces. Let $\mathcal{R}(X)$ denote the subspace spanned by the columns of $X$. Recall that $x_i, i = 1, 2, \ldots, n$, are the true low-dimensional representations of the observations. We treat the $x_i$'s as column vectors. Let $X = (x_1, x_2, \cdots, x_n)^T$; i.e., the $i$th row of $X$ corresponds to $x_i, 1 \leq i \leq n$. If the set $\{Ox_i\}$, where $O$ is a $d$ by $d$ orthogonal square matrix, forms another solution to the dimension reduction problem, we have

$$(Ox_1, Ox_2, \cdots, Ox_n)^T = XO^T.$$

It is evident that $\mathcal{R}(XO^T) = \mathcal{R}(X)$. This justifies the *invariance* that was mentioned earlier.

The goal of our performance analysis is to answer the following question: Letting $\| \tan(\cdot, \cdot) \|_2$ denote the Euclidean norm of the vector of canonical angles between two invariant subspaces ([8, Section I.5]), and letting X and $\widetilde{X}$ denote the true and estimated parameters, respectively, how do we evaluate $\| \tan(\mathcal{R}(X), \mathcal{R}(\widetilde{X})) \|_2$?

## 2.4 LTSA: Local Tangent Space Alignment

We now review LTSA. There are two main steps in the LTSA algorithm [13].

1. The first step is to compute the local representation on the manifold. Recall the projection matrix $\overline{P}_k$. It is easy to verify that $\overline{P}_k = \overline{P}_k \cdot \overline{P}_k$, which is a characteristic of projection matrices. We solve the minimization problem: $\min_{\Lambda, V} \|Y_i \overline{P}_k - \Lambda V\|_F$, where $\Lambda \in I\!\!R^{D \times d}, V \in I\!\!R^{d \times k}$, and $VV^T = I_d$. Let $V_i$ denote optimal $V$. Then the row vectors of $V_i$ are the $d$ right singular vectors of $Y_i \overline{P}_k$.

2. The solution to LTSA corresponds to the invariant subspace which is spanned and determined by the eigenvectors associated with the 2nd to the $(d + 1)$st smallest eigenvalues of the matrix

$$(S_1, \ldots, S_n)\text{diag}(\overline{P}_k - V_1^T V_1, \ldots, \overline{P}_k - V_n^T V_n)(S_1, \ldots, S_n)^T. \qquad (2)$$

where $S_i \in I\!\!R^{n \times k}$ is a selection matrix such that $Y^T S_i = Y_i$, where $Y = (y_1, y_2, \ldots, y_n)^T$.

As mentioned earlier, the subspace spanned by the eigenvectors associated with the 2nd to the $(d + 1)$st smallest eigenvalues of the matrix in 2 is an invariant subspace, which will be analyzed using matrix perturbation techniques. We slightly reformulated the original algorithm as presented in [13] for later analysis.

## 3 Perturbation Analysis

We now carry out a perturbation analysis on the reformulated version of LTSA. There are two steps: in the *local* step (Section 3.1), we characterize the deviation of the null spaces of the matrices $\overline{P}_k - V_i^T V_i, i = 1, 2, \ldots, n$. In the *global* step (Section 3.2), we derive the variation of the null space under global alignment.

### 3.1 Local Coordinates

Let $X$ be the matrix of true parameters. We define $X_i = X^T S_i = (x_1, x_2, \cdots, x_n) S_i$; i.e., the columns of $X_i$ are made by $x_i$ and those $x_j$'s that correspond to the $k - 1$ nearest neighbors of $y_i$. We require a bound on the size of the local neighborhoods defined by the $X_i$'s.

**Condition 3.1 (Universal Bound on the Sizes of Neighborhoods)** *For all $i, 1 \leq i \leq n$, we have $\tau_i < \tau$, where $\tau$ is a prescribed constant and $\tau_i$ is an upper bound on the distance between two columns of $X_i$: $\tau_i = \max_{x_j, x_k} \|x_j - x_k\|$, where the maximum is taken over all columns of $X_i$.*

In this paper, we are interested in the case when $\tau \to 0$.

We will need conditions on the local tangent spaces. Let $d_{\min,i}$ (respectively, $d_{\max,i}$) denote the minimum (respectively, maximum) singular values of $X_i \overline{P}_k$. Let

$$d_{\min} = \min_{1 \leq i \leq n} d_{\min,i}, \quad d_{\max} = \max_{1 \leq i \leq n} d_{\max,i}.$$

We can bound $d_{\max}$ as $d_{\min} \leq d_{\max} \leq \tau \sqrt{k}$ [5].

**Condition 3.2 (Local Tangent Space)** *There exists a constant $C_2 > 0$, such that*

$$C_2 \cdot \tau \leq d_{\min}. \tag{3}$$

The above can roughly be thought of as requiring that the local dimension of the manifold remain constant (i.e., the manifold has no singularities.)

The following condition defines a global bound on the errors $(\varepsilon_i)$.

**Condition 3.3 (Universal Error Bound)** *There exists $\sigma > 0$, such that $\forall i, 1 \leq i \leq n$, we have $\|y_i - f(x_i)\|_\infty < \sigma$. Moreover, we assume $\sigma = o(\tau)$; i.e., we have $\frac{\sigma}{\tau} \to 0$, as $\tau \to 0$.*

It is reasonable to require that the error bound $(\sigma)$ be smaller than the size of the neighborhood $(\tau)$, which is reflected in the above condition.

Within each neighborhood, we give a perturbation bound between an invariant subspace spanned by the true parametrization and the invariant subspace spanned by the singular vectors of the matrix of noisy observations. Let $X_i \overline{P}_k = A_i D_i B_i$ be the singular value decomposition of the matrix $X_i \overline{P}_k$; here $A_i \in I\!\!R^{d \times d}$ is orthogonal $(A_i A_i^T = I_d)$, $D_i \in I\!\!R^{d \times d}$ is diagonal, and the rows of $B_i \in I\!\!R^{d \times k}$ are the right singular vectors corresponding to the largest singular values $(B_i B_i^T = I_d)$. It is not hard to verify that

$$B_i = B_i \overline{P}_k. \tag{4}$$

Let $Y_i \overline{P}_k = \widetilde{A}_i \widetilde{D}_i \widetilde{B}_i$ be the singular value decomposition of $Y_i \overline{P}_k$, and assume that this is the "thin" decomposition of rank $d$. We may think of this as the perturbed version of $J(f; x_i^{(0)}) X_i \overline{P}_k$. The rows of $\widetilde{B}_i$ are the eigenvectors of $(Y_i \overline{P}_k)^T (Y_i \overline{P}_k)$ corresponding to the $d$ largest eigenvalues. Let $\mathcal{R}(B_i^T)$ (respectively, $\mathcal{R}(\widetilde{B}_i^T)$) denote the invariant subspace that is spanned by the columns of matrix $B_i^T$ (respectively, $\widetilde{B}_i^T$).

**Theorem 3.4** *Given invariant subspaces $\mathcal{R}(B_i^T)$ and $\mathcal{R}(\widetilde{B}_i^T))$ as defined above, we have*

$$\lim_{\tau \to 0} \| \sin(\mathcal{R}(B_i^T), \mathcal{R}(\widetilde{B}_i^T))\|_2 \leq C_3 \left( \frac{\sigma}{\tau} + C_1 \tau \right),$$

*where $C_3$ is a constant that depends on $k$, $D$ and $C_2$.*

The proof is presented in [5]. The above gives an upper bound on the deviation of the local invariant subspace in step 1 of the modified LTSA. It will be used later to prove a global upper bound.

## 3.2 Global Alignment

**Condition 3.5 (No Overuse of One Observation)** *There exists a constant $C_4$, such that*

$$\left\| \sum_{i=1}^{n} S_i \right\|_{\infty} \leq C_4.$$

Note that we must have $C_4 \geq k$. The next condition (Condition 3.6) will implicitly give an upper bound on $C_4$.

Recall that the quantity $\| \sum_{i=1}^{n} S_i \|_{\infty}$ is the maximum row sum of the absolute values of the entries in $\sum_{i=1}^{n} S_i$. The value of $\| \sum_{i=1}^{n} S_i \|_{\infty}$ is equal to the maximum number of nearest neighbor subsets to which a single observation belongs.

We will derive an upper bound on the angle between the invariant subspace spanned by the result of LTSA and the space spanned by the true parameters.

Given (4), it can be shown that $X_i \overline{P}_k (\overline{P}_k - B_i^T B_i)(X_i \overline{P}_k)^T = 0$. Recall $X = (x_1, x_2, \ldots, x_n)^T \in \mathbb{R}^{n \times d}$. It is not hard to verify that the row vectors of $(\mathbf{1}_n, X)^T$ span the $(d+1)$-dimensional null space of the matrix:

$$(S_1, \ldots, S_n)\overline{P}_k \text{diag}(I - B_1^T B_1, \ldots, I - B_n^T B_n)\overline{P}_k(S_1, \ldots, S_n)^T. \tag{5}$$

Assume that $(\frac{\mathbf{1}_n}{\sqrt{n}}, X, (X^c))^T$ is orthogonal, where $X^c \in \mathbb{R}^{n \times (n-1-d)}$. Although in our original problem formulation, we made no assumption about the $x_i$'s, we can still assume that the columns of $X$ are orthonormal because we can transform any set of $x_i$'s into an orthonormal set by rescaling the columns and multiplying by an orthogonal matrix. Based on the previous paragraph, we have

$$\begin{pmatrix} \frac{\mathbf{1}_n^T}{\sqrt{n}} \\ X^T \\ (X^c)^T \end{pmatrix} M_n \left( \frac{\mathbf{1}_n}{\sqrt{n}}, X, X^c \right) = \begin{pmatrix} \mathbf{0}_{(d+1) \times (d+1)} & \mathbf{0}_{(d+1) \times (n-d-1)} \\ \mathbf{0}_{(n-d-1) \times (d+1)} & L_2 \end{pmatrix} \tag{6}$$

where

$$M_n = (S_1, \ldots, S_n)\overline{P}_k \text{diag}(I_k - B_1^T B_1, \ldots, I_k - B_n^T B_n)\overline{P}_k(S_1, \ldots, S_n)^T$$

and

$$L_2 = (X^c)^T M_n X^c.$$

Let $\ell_{\min}$ denote the minimum singular value (i.e., eigenvalue) of $L_2$. We will need the following condition on $\ell_{\min}$.

**Condition 3.6 (Appropriateness of Global Dimension)** *$\ell_{\min} > 0$ and $\ell_{\min}$ goes to $0$ at a slower rate than $\frac{\sigma}{\tau} + \frac{1}{2}C_1\tau$; i.e., as $\tau \to 0$, we have*

$$\frac{\left( \frac{\sigma}{\tau} + \frac{1}{2}C_1\tau \right) \cdot \| \sum_{i=1}^{n} S_i \|_{\infty}}{\ell_{\min}} \to 0.$$

As discussed in [12, 11], this condition is actually related to the amount of overlap between the nearest neighbor sets.

**Theorem 3.7 (Main Theorem)**

$$\lim_{\tau \to 0} \| \tan(\mathcal{R}(\widetilde{X}), \mathcal{R}(X)) \|_2 \leq \frac{C_3(\frac{\sigma}{\tau} + C_1 \tau) \cdot \| \sum_{i=1}^{n} S_i \|_\infty}{\ell_{\min}}. \tag{7}$$

As mentioned in the Introduction, the above theorem gives a worst-case bound on the performance of LTSA. For proofs as well as a discussion of the requirement that $\sigma \to 0$ see [7]. A discussion on when Condition 3.6 is satisfied will be long and beyond the scope of this paper. We leave it to future investigation. We refer to [5] for some simulation results related to the above analysis.

## 4 A Preliminary Result on the Rate of Convergence

We discuss the rate of convergence for LTSA (to the true underlying manifold structure) in the aforementioned framework. We modify the LTSA (mainly on how to choose the size of the nearest neighborhood) for a reason that will become evident later.

We assume the following result regarding the relationship between $k$, $\ell_{\min}$, and $\tau$ (this result can be proved for $x_i$ being sampled on a uniform grid, using the properties of biharmonic eigenvalues for partial differential equations) holds:

$$\ell_{\min} \approx C(k) \cdot \nu_{\min}^{+}(\Delta^2) \cdot \tau^4, \tag{8}$$

where $\nu_{\min}^{+}(\Delta^2)$ is a constant, and $C(k) \approx k^5$. We will address such a result in the more general context in the future.

So far, we have assumed that $k$ is constant. However, allowing $k$ to be a function of the sample size $n$, say $k = n^\alpha$, where $\alpha \in [0, 1)$ allows us to control the asymptotic behavior of $\ell_{\min}$ along with the convergence of the estimated alignment matrix to the true alignment matrix.

Consider our original bound on the angle between the true coordinates and the estimated coordinates:

$$\lim_{\tau \to 0} \| \tan(\mathcal{R}(\widetilde{X}), \mathcal{R}(X)) \|_2 \leq \frac{C_3(\frac{\sigma}{\tau} + C_1 \tau) \cdot \| \sum_{i=1}^{n} S_i \|_\infty}{\ell_{\min}}.$$

Now, set $k = n^\alpha$, where $\alpha \in [0, 1)$ is an exponent, the value of which will be decided later. We must be careful in disregarding constants, since they may involve $k$. We have that $C_3 = \frac{\sqrt{kD}}{C_2}$. $C_1$ and $C_2$ are fundamental constants not involving $k$. Further, it is easy to see that $\| \sum_{i=1}^{n} S_i \|_\infty$ is $O(k)$ - since each point has $k$ neighbors, the maximum number of neighborhoods to which a point belongs is of the same order as $k$.

Now, we can use a simple heuristic to estimate the size of $\tau$, the neighborhood size. For example, suppose we fix $\epsilon$ and consider $\epsilon$-neighborhoods. For simplicity, assume that the parameter space is the unit hypercube $[0, 1]^d$, where $d$ is the intrinsic dimension. The law of large numbers tells us that $k \approx \epsilon^d \cdot n$. Thus we can approximate $\tau$ as $\tau \approx O(n^{\frac{\alpha-1}{d}})$. Plugging all this into the original equation and dropping the constants, we get

$$\lim_{\tau \to 0} \| \tan(\mathcal{R}(\widetilde{X}), \mathcal{R}(X)) \|_2 \leq \frac{n^{\frac{\alpha-1}{d}} \cdot n^{\frac{3\alpha}{2}}}{\ell_{\min}} \cdot \text{Constant}.$$

If we conjecture that the relationship in (8) holds in general (i.e., the generating coordinates can follow a more general distribution rather than only lying in a uniform grid), then we have

$$\lim_{\tau \to 0} \| \tan(\mathcal{R}(\widetilde{X}), \mathcal{R}(X)) \|_2 \leq \frac{n^{\frac{\alpha-1}{d}} \cdot n^{\frac{\alpha}{2}} \cdot n^{\alpha}}{n^{5\alpha} \cdot n^{4 \cdot \frac{\alpha-1}{d}}} \cdot \text{Constant}.$$

Now the exponent is a function only of $\alpha$ and the constant $d$. We can try to solve for $\alpha$ such that the convergence is as fast as possible. Simplifying the exponents, we get

$$\lim_{\tau \to 0} \| \tan(\mathcal{R}(\widetilde{X}), \mathcal{R}(X)) \|_2 \leq n^{\frac{-7\alpha}{2} - 3(\frac{\alpha-1}{d})} \cdot \text{Constant}.$$

As a function of $\alpha$ restricted to the interval $[0, 1)$, there is no minimum—the exponent decreases with $\alpha$, and we should choose $\alpha$ close to 1.

However, in the proof of the convergence of LTSA, it is assumed that the errors in the local step converge to 0. This error is given by

$$\| \sin(\mathcal{R}(B_i^T), \mathcal{R}(\widetilde{B}_i^T)) \|_2 \leq \frac{\sqrt{kD} \cdot [\sigma + \frac{1}{2}C_1\tau^2]}{C_2 \cdot \tau - \sqrt{kD} \cdot [\sigma + \frac{1}{2}C_1\tau^2]}.$$

Thus, our choice of $\alpha$ is restricted by the fact that the RHS of this equation must still converge to 0. Disregarding constants and writing this as a function of $n$, we get

$$\frac{n^{\frac{\alpha}{2}} \cdot n^{\frac{2\alpha-2}{d}}}{n^{\frac{\alpha-1}{d}} - n^{\frac{\alpha}{2}} \cdot n^{\frac{2\alpha-2}{d}}}.$$

This quantity converges to 0 as $n \to \infty$ if and only if we have

$$\frac{\alpha}{2} + \frac{2\alpha-2}{d} < \frac{\alpha-1}{d} \quad \Leftrightarrow \quad \alpha < \frac{2}{d+2}.$$

Note that this bound is strictly less than 1 for all positive integers $d$, so our possible choices of $\alpha$ are restricted further.

By the reasoning above, we want the exponent to be as large as possible. Further, it is easy to see that for all $d$, choosing an exponent roughly equal to $\frac{2}{d+2}$ will always yield a bound converging to 0. The following table gives the optimal exponents for selected values of $d$ along with the convergence rate of $\lim_{\tau \to 0} \| \tan(\mathcal{R}(\widetilde{X}), \mathcal{R}(X)) \|_2$. In general, using the optimal value of $\alpha$, the *convergence rate* will be roughly $n^{\frac{-4}{d+2}}$.

Table 1: Convergence rates for a few values of the underlying dimension $d$.

| $d$ | 1 | 2 | 3 | 4 | 5 |
|---|---|---|---|---|---|
| Optimal $\alpha$ | 0.66 | 0.5 | 0.4 | 0.33 | 0.29 |
| Convergence rate | $-1.33$ | $-1$ | $-0.8$ | $-0.66$ | $-0.57$ |

Thesis [7] presents some numerical experiments to illustrate the above results. Associated with each fixed value of $k$, there seems to be a threshold value of $n$, above which the performance degrades. This value increases with $k$, though perhaps at the cost of worse performance for small $n$. However, we expect from the above analysis that, regardless of the value chosen, the performance will eventually become unacceptable for any fixed $k$.

## 5   Discussion

To the best of our knowledge, the performance analysis that is based on invariant subspaces is new. Consequently the worst-case upper bound is the first of its kind. There are still open questions to be addressed (Section 5.1). In addition to a discussion on the relation of LTSA to existing dimension reduction methodologies, we will also address relation with known results as well (Section 5.2).

### 5.1   Open Questions

The rate of convergence of $\ell_{\min}$ is determined by the topological structure of $f$. It is important to estimate this rate of convergence, but this issue has not been addressed here. We did not address the correctness of (8) at all. It turns out the proof of (8) is quite nontrivial and tedious.

We assume that $\tau \to 0$. One can imagine that it is true when the error bound ($\sigma$) goes to 0 and when the $x_i$'s are sampled with a sufficient density in the support of $f$. An open problem is how to derive the rate of convergence of $\tau \to 0$ as a function of the topology of $f$ and the sampling scheme. After doing so, we may be able to decide where our theorem is applicable.

### 5.2   Relation to Existing Work

The error analysis in the original paper about LTSA is the closest to our result. However, Zhang and Zha [13] do not interpret their solutions as invariant subspaces, and hence their analysis does not yield a worst case bound as we have derived here.

Reviewing the original papers on LLE [6], Laplacian eigenmaps [1], and Hessian eigenmaps [3] reveals that their solutions are subspaces spanned by a specific set of eigenvectors. This naturally suggests that results analogous to ours may be derivable as well for these algorithms. A recent book chapter [4] stresses this point. After deriving corresponding upper bounds, we can establish different proofs of consistency than those presented in these papers.

ISOMAP, another popular manifold learning algorithm, is an exception. Its solution cannot immediately be rendered as an invariant subspace. However, ISOMAP calls for MDS, which can be associated with an invariant subspace; one may derive an analytical result through this route.

## 6  Conclusion

We derive an upper bound of the distance between two invariant subspaces that are associated with the numerical output of LTSA and an assumed intrinsic parametrization. Such a bound describes the performance of LTSA with errors in the observations, and thus creates a theoretical foundation for its use in real-world applications in which we would naturally expect such errors to be present. Our results can also be used to show other desirable properties, including consistency and rate of convergence. Similar bounds may be derivable for other manifold-based learning algorithms.

## References

[1] M. Belkin and P. Niyogi. Laplacian eigenmaps for dimensionality reduction and data representation. *Neural Computation*, 15(6):1373–1396, 2003.

[2] M. Brand. Charting a manifold. In *Neural Information Processing Systems*, volume 15. Mitsubishi Electric Research Labs, MIT Press, March 2003.

[3] D. L. Donoho and C. E. Grimes. Hessian eigenmaps: New locally linear embedding techniques for high-dimensional data. *Proceedings of the National Academy of Arts and Sciences*, 100:5591–5596, 2003.

[4] X. Huo, X. S. Ni, and A. K. Smith. *Mining of Enterprise Data*, chapter A survey of manifold-based learning methods. Springer, New York, 2005. Invited book chapter, accepted.

[5] X. Huo and A. K. Smith. Performance analysis of a manifold learning algorithm in dimension reduction. Technical report, Georgia Institute of Technology, March 2006. Downloadable at www2.isye.gatech.edu/statistics/papers/06-06.pdf, to appear in Linear Algebra and Its Applications.

[6] S. T. Roweis and L. K. Saul. Nonlinear dimensionality reduction by locally linear embedding. *Science*, 290:2323–2326, 2000.

[7] A. K. Smith. New results in dimension reduction and model selection. Ph.D. Thesis. Available at http://etd.gatech.edu, 2008.

[8] G. W. Stewart and J.-G. Sun. *Matrix Perturbation Theory*. Academic Press, Boston, MA, 1990.

[9] J. B. Tenenbaum, V. de Silva, and J. C. Langford. A global geometric framework for nonlinear dimensionality reduction. *Science*, 290:2319–2323, 2000.

[10] T. Wittman. MANIfold learning Matlab demo. URL: http://www.math.umn.edu/~wittman/mani/index.html, April 2005.

[11] H. Zha and H. Zhang. Spectral properties of the alignment matrices in manifold learning. *SIAM Review*, 2008.

[12] H. Zha and Z. Zhang. Spectral analysis of alignment in manifold learning. In *Proceedings of IEEE International Conference on Acoustics, Speech, and Signal Processing*, 2005.

[13] Z. Zhang and H. Zha. Principal manifolds and nonlinear dimension reduction via local tangent space alignment. *SIAM Journal of Scientific Computing*, 26(1):313–338, 2004.

